# Particle-based Variational Inference for Continuous Systems

**Alexander T. Ihler**
Dept. of Computer Science
Univ. of California, Irvine
ihler@ics.uci.edu

**Andrew J. Frank**
Dept. of Computer Science
Univ. of California, Irvine
ajfrank@ics.uci.edu

**Padhraic Smyth**
Dept. of Computer Science
Univ. of California, Irvine
smyth@ics.uci.edu

## Abstract

Since the development of loopy belief propagation, there has been considerable work on advancing the state of the art for approximate inference over distributions defined on discrete random variables. Improvements include guarantees of convergence, approximations that are provably more accurate, and bounds on the results of exact inference. However, extending these methods to continuous-valued systems has lagged behind. While several methods have been developed to use belief propagation on systems with continuous values, recent advances for discrete variables have not as yet been incorporated.

In this context we extend a recently proposed particle-based belief propagation algorithm to provide a general framework for adapting discrete message-passing algorithms to inference in continuous systems. The resulting algorithms behave similarly to their purely discrete counterparts, extending the benefits of these more advanced inference techniques to the continuous domain.

## 1   Introduction

Graphical models have proven themselves to be an effective tool for representing the underlying structure of probability distributions and organizing the computations required for exact and approximate inference. Early examples of the use of graph structure for inference include join or junction trees [1] for exact inference, Markov chain Monte Carlo (MCMC) methods [2], and variational methods such as mean field and structured mean field approaches [3]. Belief propagation (BP), originally proposed by Pearl [1], has gained in popularity as a method of approximate inference, and in the last decade has led to a number of more sophisticated algorithms based on conjugate dual formulations and free energy approximations [4, 5, 6].

However, the progress on approximate inference in systems with continuous random variables has not kept pace with that for discrete random variables. Some methods, such as MCMC techniques, are directly applicable to continuous domains, while others such as belief propagation have approximate continuous formulations [7, 8]. Sample-based representations, such as are used in particle filtering, are particularly appealing as they are relatively easy to implement, have few numerical issues, and have no inherent distributional assumptions. Our aim is to extend particle methods to take advantage of recent advances in approximate inference algorithms for discrete-valued systems.

Several recent algorithms provide significant advantages over loopy belief propagation. Double-loop algorithms such as CCCP [9] and UPS [10] use the same approximations as BP but guarantee convergence. More general approximations can be used to provide theoretical bounds on the results of exact inference [5, 3] or are guaranteed to improve the quality of approximation [6], allowing an informed trade-off between computation and accuracy. Like belief propagation, they can be formulated as local message-passing algorithms on the graph, making them amenable to parallel computation [11] or inference in distributed systems [12, 13].

In short, the algorithmic characteristics of these recently-developed algorithms are often better, or at least more flexible, than those of BP. However, these methods have not been applied to continuous random variables, and in fact this subject was one of the open questions posed at a recent NIPS workshop [14].

In order to develop particle-based approximations for these algorithms, we focus on one particular technique for concreteness: tree-reweighted belief propagation (TRW) [5]. TRW represents one of the earliest of a recent class of inference algorithms for discrete systems, but as we discuss in Section 2.2 the extensions of TRW can be incorporated into the same framework if desired.

The basic idea of our algorithm is simple and extends previous particle formulations of exact inference [15] and loopy belief propagation [16]. We use collections of samples drawn from the continuous state space of each variable to define a discrete problem, "lifting" the inference task from the original space to a restricted, discrete domain on which TRW can be performed. At any point, the current results of the discrete inference can be used to re-select the sample points from a variable's continuous domain. This iterative interaction between the sample locations and the discrete messages produces a dynamic discretization that adapts itself to the inference results.

We demonstrate that TRW and similar methods can be naturally incorporated into the lifted, discrete phase of particle belief propagation and that they confer similar benefits on the continuous problem as hold in truly discrete systems. To this end we measure the performance of the algorithm on an Ising grid, an analogous continuous model, and the sensor localization problem. In each case, we show that tree-reweighted particle BP exhibits behavior similar to TRW and produces significantly more robust marginal estimates than ordinary particle BP.

## 2 Graphical Models and Inference

Graphical models provide a convenient formalism for describing structure within a probability distribution $p(X)$ defined over a set of variables $X = \{x_1, \ldots, x_n\}$. This structure can then be applied to organize computations over $p(X)$ and construct efficient algorithms for many inference tasks, including optimization to find a maximum a posteriori (MAP) configuration, marginalization, or computing the likelihood of observed data.

### 2.1 Factor Graphs

Factor graphs [17] are a particular type of graphical model that describe the factorization structure of the distribution $p(X)$ using a bipartite graph consisting of *factor* nodes and *variable* nodes. Specifically, suppose such a graph $G$ consists of factor nodes $F = \{f_1, \ldots, f_m\}$ and variable nodes $X = \{x_1, \ldots, x_n\}$. Let $X_u \subseteq X$ denote the neighbors of factor node $f_u$ and $F_s \subseteq F$ denote the neighbors of variable node $x_s$. Then, $G$ is consistent with a distribution $p(X)$ if and only if

$$p(x_1, \ldots, x_n) = \frac{1}{Z} \prod_{u=1}^{m} f_u(X_u). \tag{1}$$

In a common abuse of notation, we use the same symbols to represent each variable node and its associated variable $x_s$, and similarly for each factor node and its associated function $f_u$. Each factor $f_u$ corresponds to a strictly positive function over a subset of the variables. The graph connectivity captures the conditional independence structure of $p(X)$, enabling the development of efficient exact and approximate inference algorithms [1, 17, 18]. The quantity $Z$, called the partition function, is also of importance in many problems; for example in normalized distributions such as Bayes nets, it corresponds to the probability of evidence and can be used for model comparison.

A common inference problem is that of computing the marginal distributions of $p(X)$. Specifically, for each variable $x_s$ we are interested in computing the marginal distribution

$$p_s(x_s) = \int_{X \setminus x_s} p(X) \, \partial X.$$

For discrete-valued variables $X$, the integral is replaced by a summation.

When the variables are discrete and the graph $G$ representing $p(X)$ forms a tree ($G$ has no cycles), marginalization can be performed efficiently using the belief propagation or sum-product algorithm [1, 17]. For inference in more general graphs, the junction tree algorithm [19] creates a

tree-structured hypergraph of $G$ and then performs inference on this hypergraph. The computational complexity of this process is $O(nd^b)$, where $d$ is the number of possible values for each variable and $b$ is the maximal clique size of the hypergraph. Unfortunately, for even moderate values of $d$, this complexity becomes prohibitive for even relatively small $b$.

## 2.2 Approximate Inference

Loopy BP [1] is a popular alternative to exact methods and proceeds by iteratively passing "messages" between variable and factor nodes in the graph as though the graph were a tree (ignoring cycles). The algorithm is exact when the graph is tree-structured and can provide excellent approximations in some cases even when the graph has loops. However, in other cases loopy BP may perform poorly, have multiple fixed points, or fail to converge at all.

Many of the more recent varieties of approximate inference are framed explicitly as an optimization of local approximations over locally defined cost functions. Variational or free-energy based approaches convert the problem of exact inference into the optimization of a free energy function over the set of realizable marginal distributions $\mathcal{M}$, called the *marginal polytope* [18]. Approximate inference then corresponds to approximating the constraint set and/or energy function. Formally,

$$\max_{\boldsymbol{\mu} \in \mathcal{M}} \mathbb{E}_{\boldsymbol{\mu}}[\log P(X)] + \mathbb{H}(\boldsymbol{\mu}) \approx \max_{\boldsymbol{\mu} \in \widehat{\mathcal{M}}} \mathbb{E}_{\boldsymbol{\mu}}[\log P(X)] + \widehat{\mathbb{H}}(\boldsymbol{\mu})$$

where $\mathbb{H}$ is the entropy of the distribution corresponding to $\boldsymbol{\mu}$. Since the solution $\boldsymbol{\mu}$ may not correspond to the marginals of any consistent joint distribution, these approximate marginals are typically referred to as *pseudomarginals*. If both the constraints in $\widehat{\mathcal{M}}$ and approximate entropy $\widehat{\mathbb{H}}$ decompose locally on the graph, the optimization process can be interpreted as a message-passing procedure, and is often performed using fixed-point equations like those of BP.

Belief propagation can be understood in this framework as corresponding to an outer approximation $\widehat{\mathcal{M}} \supseteq \mathcal{M}$ enforcing local consistency and the Bethe approximation to $\mathbb{H}$ [4]. This viewpoint provides a clear path to directly improve upon the properties of BP, leading to a number of different algorithms. For example, CCCP [9] and UPS [10] make the same approximations but use an alternative, direct optimization procedure to ensure convergence. Fractional belief propagation [20] corresponds to a more general Bethe-like approximation with additional parameters, which can be modified to ensure that the cost function is convex and used with convergent algorithms [21]. A special case includes tree-reweighted belief propagation [5], which both ensures convexity and provides an upper bound on the partition function $Z$. The approximation of $\mathcal{M}$ can also be improved using cutting plane methods, which include additional, higher-order consistency constraints on the pseudomarginals [6]. Other choices of local cost functions lead to alternative families of approximations [8].

Overall, these advances have provided significant improvements in the state of the art for approximate inference in discrete-valued systems. They provide increased flexibility, theoretical bounds on the results of exact inference, and can provably increase the quality of the estimates. However, these advances have not been carried over into the continuous domain.

For concreteness, in the rest of the paper we will use tree-reweighted belief propagation (TRW) [5] as our inference method of choice, although the same ideas can be applied to any of the discussed inference algorithms. As we will see shortly, the details specific to TRW are nicely encapsulated and can be swapped out for those of another algorithm with minimal effort.

The fixed-point equations for TRW lead to a message-passing algorithm similar to BP, defined by

$$m_{x_s \to f_u}(x_s) \propto \prod_{f_v \in F_s} \frac{m_{f_v \to x_s}(x_s)^{\rho_v}}{m_{f_u \to x_s}(x_s)} \quad , \quad m_{f_u \to x_s}(x_s) \propto \sum_{X_u \setminus x_s} f_u(X_u)^{1/\rho_u} \prod_{x_t \in X_u \setminus x_s} m_{x_t \to f_u}(x_t)$$

$$(2)$$

The parameters $\rho_v$ are called edge weights or appearance probabilities. For TRW, the $\rho$ are required to correspond to the fractional occurrence rates of the edges in some collection of tree-structured subgraphs of $G$. The choice of $\rho$ affects the quality of the approximation; the tightest upper bound can be obtained via a convex optimization of $\rho$ which computes the pseudomarginals as an inner loop.

# 3 Continuous Random Variables

For continuous-valued random variables, many of these algorithms cannot be applied directly. In particular, any reasonably fine-grained discretization produces a discrete variable whose domain size $d$ is quite large. The domain size is typically exponential in the dimension of the variable and the complexity of the message-passing algorithms is $O(nd^b)$, where $n$ is the total number of variables and $b$ is the number of variables in the largest factor. Thus, the computational cost can quickly become intractable even with pairwise factors over low dimensional variables. Our goal is to adapt the algorithms of Section 2.2 to perform efficient approximate inference in such systems.

For time-series problems, in which $G$ forms a chain, a classical solution is to use sequential Monte Carlo approximations, generally referred to as particle filtering [22]. These methods use samples to define an adaptive discretization of the problem with fine granularity in regions of high probability. The stochastic nature of the discretization is simple to implement and enables probabilistic assurances of quality including convergence rates which are independent of the problem's dimensionality. (In sufficiently few dimensions, deterministic adaptive discretizations can also provide a competitive alternative, particularly if the factors are analytically tractable [23, 24].)

## 3.1 Particle Representations for Message-Passing

Particle-based approximations have been extended to loopy belief propagation as well. For example, in the nonparametric belief propagation (NBP) algorithm [7], the BP messages are represented as Gaussian mixtures and message products are approximated by drawing samples, which are then smoothed to form new Gaussian mixture distributions. A key aspect of this approach is the fact that the product of several mixtures of Gaussians is also a mixture of Gaussians, and thus can be sampled from with relative ease. However, it is difficult to see how to extend this algorithm to more general message-passing algorithms, since for example the TRW fixed point equations (2) involve ratios and powers of messages, which do not have a simple form for Gaussian mixtures and may not even form finitely integrable functions.

Instead, we adapt a recent particle belief propagation (PBP) algorithm [16] to work on the tree-reweighted formulation. In PBP, samples (particles) are drawn for each variable, and each message is represented as a set of weights over the available values of the target variable. At a high level, the procedure iterates between sampling particles from each variable's domain, performing inference over the resulting discrete problem, and adaptively updating the sampling distributions. This process is illustrated in Figure 1. Formally, we define a proposal distribution $W_s(x_s)$ for each variable $x_s$ such that $W_s(x_s)$ is non-zero over the domain of $x_s$. Note that we may rewrite the factor message computation (2) as an importance reweighted expectation:

$$m_{f_u \to x_s}(x_s) \propto \mathop{\mathbb{E}}_{X_u \setminus x_s} \left[ f_u(X_u)^{1/\rho_u} \prod_{x_t \in X_u \setminus x_s} \frac{m_{x_t \to f_u}(x_t)}{W_t(x_t)} \right] \tag{3}$$

Let us index the variables that are neighbors of factor $f_u$ as $X_u = \{x_{u_1}, \ldots, x_{u_b}\}$. Then, after sampling particles $\{x_s^{(1)}, \cdots, x_s^{(N)}\}$ from $W_s(x_s)$, we can index a particular assignment of particle values to the variables in $X_u$ with $X_u^{(\vec{j})} = [x_{u_1}^{(j_1)}, \ldots, x_{u_b}^{(j_b)}]$. We then obtain a finite-sample approximation of the factor message in the form

$$m_{f_u \to x_{u_k}}\left(x_{u_k}^{(j)}\right) \propto \frac{1}{N^{b-1}} \sum_{\vec{i}:i_k=j} \left[ f_u\left(X_u^{(\vec{i})}\right)^{1/\rho_u} \prod_{l \neq k} \frac{m_{x_{u_l} \to f_u}\left(x_{u_l}^{(i_l)}\right)}{W_{x_{u_l}}\left(x_{u_l}^{(i_l)}\right)} \right] \tag{4}$$

In other words, we construct a Monte Carlo approximation to the integral using importance weighted samples from the proposal. Each of the values in the message then represents an estimate of the continuous function (2) evaluated at a single particle. Observe that the sum is over $N^{b-1}$ elements, and hence the complexity of computing an entire factor message is $O(N^b)$; this could be made more efficient at the price of increased stochasticity by summing over a random subsample of the vectors

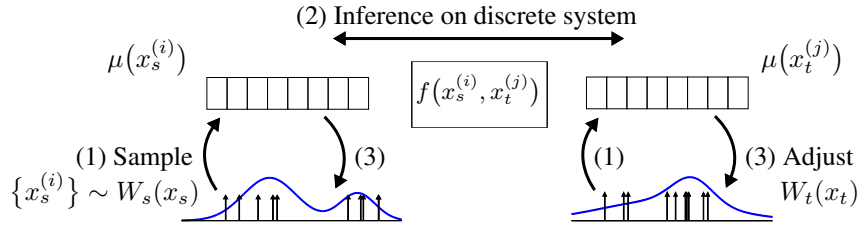

Figure 1: Schematic view of particle-based inference. (1) Samples for each variable provide a dynamic discretization of the continuous space; (2) inference proceeds by optimization or message-passing in the discrete space; (3) the resulting local functions can be used to change the proposals $W_s(\cdot)$ and choose new sample locations for each variable.

$\vec{i}$. Likewise, we compute variable messages and beliefs as simple point-wise products:

$$m_{x_s \to f_u}\left(x_s^{(j)}\right) \propto \frac{\prod_{f_v \in F_s} m_{f_v \to x_s}\left(x_s^{(j)}\right)^{\rho_v}}{m_{f_u \to x_s}\left(x_s^{(j)}\right)} \quad , \quad b_s(x_s^{(j)}) \propto \prod_{f_v \in F_s} m_{f_v \to x_s}\left(x_s^{(j)}\right)^{\rho_v} \quad (5)$$

This parallels the development in [16], except here we use factor weights $\vec{\rho}$ to compute messages according to TRW rather than standard loopy BP.

Just as in discrete problems, it is often desirable to obtain estimates of the log partition function for use in goodness-of-fit testing or model comparison. Our implementation of TRW-PBP gives us a stochastic estimate of an upper bound on the true partition function. Using other message passing approaches that fit into this framework, such as mean field, can provide a similar a lower bound. These bounds provide a possible alternative to Monte Carlo estimates of marginal likelihood [25].

## 3.2   Rao-Blackwellized Estimates

Quantities about $x_s$ such as expected values under the pseudomarginal can be computed using the samples $x_s^{(i)}$. However, for any given variable node $x_s$, the incoming messages to $x_s$ given in (4) are defined in terms of the importance weights and sampled values of the neighboring variables. Thus, we can compute an estimate of the messages and beliefs defined in (4)–(5) at arbitrary values of $x_s$, simply by evaluating (4) at that point. This allows us to perform *Rao-Blackwellization*, conditioning on the samples at the neighbors of $x_s$ rather than using $x_s$'s samples directly.

Using this trick we can often get much higher quality estimates from the inference for small $N$. In particular, if the variable state spaces are sufficiently small that they can be discretized (for example, in 3 or fewer dimensions the discretized domain size $d$ may be manageable) but the resulting factor domain size, $d^b$, is intractably large, we can evaluate (4) on the discretized grid for only $O(dN^{b-1})$. More generally, we can substitute a larger number of samples $N' \gg N$ with cost that grows only linearly in $N'$.

## 3.3   Resampling and Proposal Distributions

Another critical point is that the efficiency of this procedure hinges on the quality of the proposal distributions $W_s$. Unfortunately, this forms a circular problem – $W$ must be chosen to perform inference, but the quality of $W$ depends on the distribution and its pseudomarginals. This interdependence motivates an attempt to learn the sampling distributions in an online fashion, adaptively updating them based on the results of the partially completed inference procedure. Note that this procedure depends on the same properties as Rao-Blackwellized estimates: that we be able to compute our messages and beliefs at a new set of points given the message weights at the other nodes.

Both [15] and [16] suggest using the current belief at each iteration to form a new proposal distribution. In [15], parametric density estimates are formed using the message-weighted samples at the current iteration, which form the sampling distributions for the next phase. In [16], a short Metropolis-Hastings MCMC sequence is run at a single node, using the Rao-Blackwellized belief estimate to compute an acceptance probability. A third possibility is to use a sampling/importance

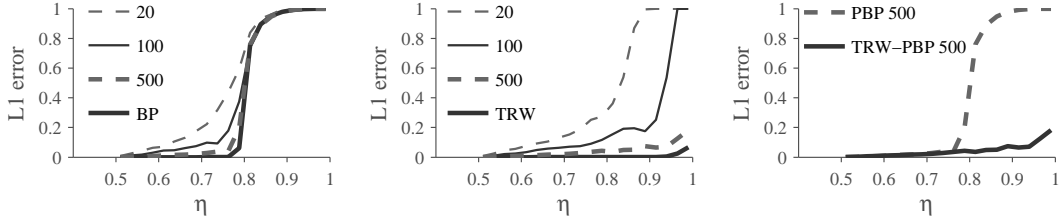

Figure 2: 2-D Ising model performance. L1 error for PBP (left) and TRW-PBP (center) for varying numbers of particles; (right) PBP and TRW-PBP juxtaposed to reveal the gap for high $\eta$.

resampling (SIR) procedure, drawing a large number of samples, computing weights, and probabilistically retaining only $N$. In our experiments we draw samples from the current beliefs, as approximated by Rao-Blackwellized estimation over a fine grid of particles. For variables in more than 2 dimensions, we recommend the Metropolis-Hastings approach.

# 4    Ising-like Models

The Ising model corresponds to a graphical model, typically a grid, over binary-valued variables with pairwise factors. Originating in statistical physics, similar models are common in many applications including image denoising and stereo depth estimation. Ising models are well understood, and provide a simple example of how BP can fail and the benefits of more general forms such as TRW. We initially demonstrate the behavior of our particle-based algorithms on a small $(3 \times 3)$ lattice of binary-valued variables to compare with the exact discrete implementations, then show that the same observed behavior arises in an analagous continuous-valued problem.

## 4.1    Ising model

Our factors consist of single-variable and pairwise functions, given by

$$f(x_s) = [ \begin{array}{cc} 0.5 & 0.5 \end{array} ] \qquad\qquad f(x_s, x_t) = \left[ \begin{array}{cc} \eta & 1 - \eta \\ 1 - \eta & \eta \end{array} \right] \qquad (6)$$

for $\eta > .5$. By symmetry, it is easy to see that the true marginal of each variable is uniform, $[.5 \ .5]$. However, around $\eta \approx .78$ there is a phase transition; the uniform fixed point becomes unstable and several others appear, becoming more skewed toward one state or another as $\eta$ increases. As the strength of coupling in an Ising model increases, the performance of BP often degrades sharply, while TRW is comparatively robust and remains near the true marginals [5].

Figure 2 shows the performance of PBP and TRW-PBP on this model. Each data point represents the median $L_1$ error between the beliefs and the true marginals, across all nodes and 40 randomly initialized trials, after 50 iterations. The left plot (BP) clearly shows the phase shift; in contrast, the error of TRW remains low even for very strong interactions. In both cases, as $N$ increases the particle versions of the algorithms converge to their discrete equivalents.

## 4.2    Continuous grid model

The results for discrete systems, and their corresponding intuition, carry over naturally into continuous systems as well. To illustrate on an interpretable analogue of the Ising model, we use the same graph structure but with real-valued variables, and factors given by:

$$f(x_s) = \exp\left(-\frac{x_s^2}{2\sigma_l^2}\right) + \exp\left(-\frac{(x_s - 1)^2}{2\sigma_l^2}\right) \qquad f(x_s, x_t) = \exp\left(-\frac{|x_s - x_t|^2}{2\sigma_p^2}\right). \qquad (7)$$

Local factors consist of bimodal Gaussian mixtures centered at 0 and 1, while pairwise factors encourage similarity using a zero-mean Gaussian on the distance between neighboring variables. We set $\sigma_l = 0.2$ and vary $\sigma_p$ analogously to $\eta$ in the discrete model. Since all potentials are Gaussian mixtures, the joint distribution is also a Gaussian mixture and can be computed exactly.

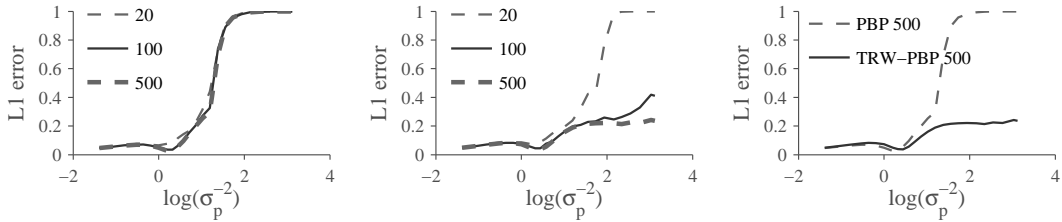

Figure 3: Continuous grid model performance. L1 error for PBP (left) and TRW-PBP (center) for varying numbers of particles; (right) PBP and TRW-PBP juxtaposed to reveal the gap for low $\sigma_p$.

Figure 3 shows the results of running PBP and TRW-PBP on the continuous grid model, demonstrating similar characteristics to the discrete model. The left panel reveals that our continuous grid model also induces a phase shift in PBP, much like that of the Ising model. For sufficiently small values of $\sigma_p$ (large values on our transformed axis), the beliefs in PBP collapse to unimodal distributions with an $L_1$ error of 1. In contrast, TRW-PBP avoids this collapse and maintains multi-modal distributions throughout; its primary source of error (0.2 at 500 particles) corresponds to overdispersed bimodal beliefs. This is expected in attractive models, in which BP tends to "overcount" information leading to underestimates of variance; TRW removes some of this overcounting and may overestimate uncertainty.

As mentioned in Section 3.1, we can use the results of TRW-PBP to compute an upper bound on the log partition function. We implement naive mean field within this same framework to achieve a lower bound as well. The resulting bounds, computed for a continuous grid model in which mean field collapses to a single mode, are shown in Figure 4. With sufficiently many particles, the values produced by TRW-PBP and MF inference bound the true value, as they should. With only 20 particles per variable, however, TRW-PBP occasionally fails and yields "upper bounds" below the true value. This is not surprising; the consistency guarantees associated with the importance-reweighted expectation take effect only when $N$ is sufficiently large.

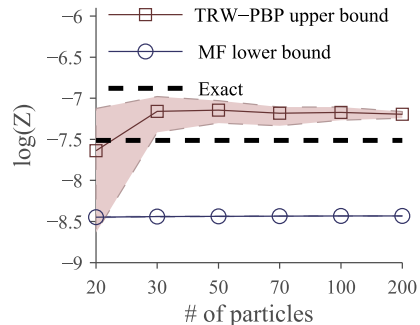

Figure 4: Bounds on the log partition function.

## 5 Sensor Localization

We also demonstrate the presence of these effects in a simulation of a real-world application. *Sensor localization* considers the task of estimating the position of a collection of sensors in a network given noisy estimates of a subset of the distances between pairs of sensors, along with known positions for a small number of *anchor nodes*. Typical localization algorithms operate by optimizing to find the most likely joint configuration of sensor positions. A classical model consists of (at a minimum) three anchor nodes, and a Gaussian model on the noise in the distance observations.

In [12], this problem is formulated as a graphical model and an alternative solution is proposed using nonparametric belief propagation to perform approximate marginalization. A significant advantage of this approach is that by providing approximate marginals, we can estimate the degree of uncertainty in the sensor positions. Gauging this uncertainty can be particularly important when the distance information is sufficiently ambiguous that the posterior belief is multi-modal, since in this case the estimated sensor position may be quite far from its true value. Unfortunately, belief propagation is not ideal for identifying multimodality, since the model is essentially attractive. BP may underestimate the degree of uncertainty in the marginal distributions and (as in the case of the Ising-like models in the previous section) collapse into a single mode, providing beliefs which are misleadingly overconfident.

Figure 5 shows a set of sensor configurations where this is the case. The distance observations induce a fully connected graph; the edges are omitted for clarity. In this network the anchor nodes are nearly collinear. This induces a bimodal uncertainty about the locations of the remaining nodes

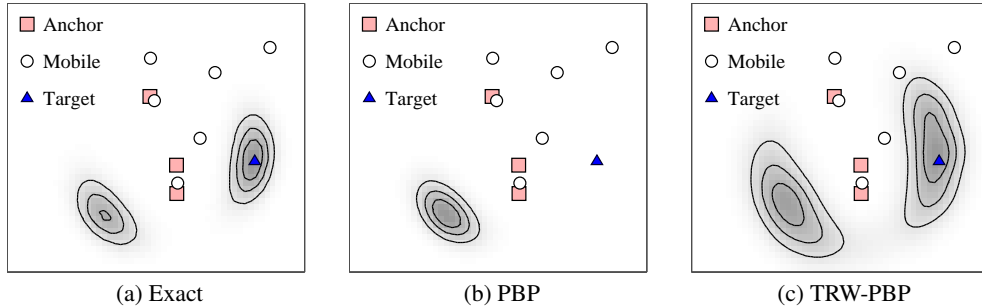

| (a) Exact | (b) PBP | (c) TRW-PBP |

Figure 5: Sensor location belief at the target node. (a) Exact belief computed using importance sampling. (b) PBP collapses and represents only one of the two modes. (c) TRW-PBP overestimates the uncertainty around each mode, but represents both.

– the configuration in which they are all reflected across the crooked line formed by the anchors is nearly as likely as the true configuration. Although this example is anecdotal, it reflects a situation which can arise regularly in practice [26].

Figure 5a shows the true marginal distribution for one node, estimated exhaustively using importance sampling with $5 \times 10^6$ samples. It shows a clear bimodal structure – a slightly larger mode near the sensor's true location and a smaller mode at a point corresponding to the reflection. In this system there is not enough information in the measurements to resolve the sensor positions. We compare these marginals to the results found using PBP.

Figure 5b displays the Rao-Blackwellized belief estimate for one node after 20 iterations of PBP with each variable represented by 100 particles. Only one mode is present, suggesting that PBP's beliefs have "collapsed," just as in the highly attractive Ising model. Examination of the other nodes' beliefs (not shown for space) confirms that all are unimodal distributions centered around their reflected locations. It is worth noting that PBP converged to the alternative set of unimodal beliefs (supporting the true locations) in about half of our trials. Such an outcome is only slightly better; an accurate estimate of confidence is equally important.

The corresponding belief estimate generated by TRW-PBP is shown in Figure 5c. It is clearly bimodal, with significant probability mass supporting both the true and reflected locations. Also, each of the two modes is less concentrated than the belief in 5b. As with the continuous grid model we see increased stability at the price of conservative overdispersion. Again, similar effects occur for the other nodes in the network.

# 6 Conclusion

We propose a framework for extending recent advances in discrete approximate inference for application to continuous systems. The framework directly integrates reweighted message passing algorithms such as TRW into the lifted, discrete phase of PBP. Furthermore, it allows us to iteratively adjust the proposal distributions, providing a discretization that adapts to the results of inference, and allows us to use Rao-Blackwellized estimates to improve our final belief estimates.

We consider the particular case of TRW and show that its benefits carry over directly to continuous problems. Using an Ising-like system, we argue that phase transitions exist for particle versions of BP similar to those found in discrete systems, and that TRW significantly improves the quality of the estimate in those regimes. This improvement is highly relevant to approximate marginalization for sensor localization tasks, in which it is important to accurately represent the posterior uncertainty.

The flexibility in the choice of message passing algorithm makes it easy to consider several instantiations of the framework and use the one best suited to a particular problem. Furthermore, future improvements in message-passing inference algorithms on discrete systems can be directly incorporated into continuous problems.

**Acknowledgements:** This material is based upon work partially supported by the Office of Naval Research under MURI grant N00014-08-1-1015.

## References

[1] J. Pearl. *Probabilistic Reasoning in Intelligent Systems*. Morgan Kaufman, San Mateo, 1988.

[2] S. Geman and D. Geman. Stochastic relaxation, Gibbs distributions, and the Bayesian restoration of images. *IEEE Trans. PAMI*, 6(6):721–741, November 1984.

[3] M. Jordan, Z. Ghahramani, T. Jaakkola, and L. Saul. An introduction to variational methods for graphical methods. *Machine Learning*, 37:183–233, 1999.

[4] J. Yedidia, W. Freeman, and Y. Weiss. Constructing free energy approximations and generalized belief propagation algorithms. Technical Report 2004-040, MERL, May 2004.

[5] M. Wainwright, T. Jaakkola, and A. Willsky. A new class of upper bounds on the log partition function. *IEEE Trans. Info. Theory*, 51(7):2313–2335, July 2005.

[6] D. Sontag and T. Jaakkola. New outer bounds on the marginal polytope. In *NIPS 20*, pages 1393–1400. MIT Press, Cambridge, MA, 2008.

[7] E. Sudderth, A. Ihler, W. Freeman, and A. Willsky. Nonparametric belief propagation. In *CVPR*, 2003.

[8] T. Minka. Divergence measures and message passing. Technical Report 2005-173, Microsoft Research Ltd, January 2005.

[9] A. Yuille. CCCP algorithms to minimize the Bethe and Kikuchi free energies: convergent alternatives to belief propagation. *Neural Comput.*, 14(7):1691–1722, 2002.

[10] Y.-W. Teh and M. Welling. The unified propagation and scaling algorithm. In *NIPS 14*. 2002.

[11] J. Gonzalez, Y. Low, and C. Guestrin. Residual splash for optimally parallelizing belief propagation. In *In Artificial Intelligence and Statistics (AISTATS)*, Clearwater Beach, Florida, April 2009.

[12] A. Ihler, J. Fisher, R. Moses, and A. Willsky. Nonparametric belief propagation for self-calibration in sensor networks. *IEEE J. Select. Areas Commun.*, pages 809–819, April 2005.

[13] J. Schiff, D. Antonelli, A. Dimakis, D. Chu, and M. Wainwright. Robust message-passing for statistical inference in sensor networks. In *IPSN*, pages 109–118, April 2007.

[14] A. Globerson, D. Sontag, and T. Jaakkola. *Approximate inference – How far have we come? (NIPS'08 Workshop)*, 2008. http://www.cs.huji.ac.il/~gamir/inference-workshop.html.

[15] D. Koller, U. Lerner, and D. Angelov. A general algorithm for approximate inference and its application to hybrid Bayes nets. In *UAI 15*, pages 324–333, 1999.

[16] A. Ihler and D. McAllester. Particle belief propagation. In *AI & Statistics: JMLR W&CP*, volume 5, pages 256–263, April 2009.

[17] F. Kschischang, B. Frey, and H.-A. Loeliger. Factor graphs and the sum-product algorithm. *IEEE Trans. Info. Theory*, 47(2):498–519, February 2001.

[18] M. Wainwright and M. Jordan. Graphical models, exponential families, and variational inference. Technical Report 629, UC Berkeley Dept. of Statistics, September 2003.

[19] SL Lauritzen and DJ Spiegelhalter. Local computations with probabilities on graphical structures and their application to expert systems. *Journal of the Royal Statistical Society. Series B (Methodological)*, pages 157–224, 1988.

[20] W. Wiegerinck and T. Heskes. Fractional belief propagation. In *NIPS 15*, pages 438–445. 2003.

[21] T. Hazan and A. Shashua. Convergent message-passing algorithms for inference over general graphs with convex free energies. In *UAI 24*, pages 264–273. July 2008.

[22] M. S. Arulampalam, S. Maskell, N. Gordon, and T. Clapp. A tutorial on particle filters for online nonlinear/non-Gaussian Bayesian tracking. 50(2):174–188, February 2002.

[23] J. Coughlan and H. Shen. Dynamic quantization for belief propagation in sparse spaces. *Comput. Vis. Image Underst.*, 106(1):47–58, 2007.

[24] M. Isard, J. MacCormick, and K. Achan. Continuously-adaptive discretization for message-passing algorithms. In *NIPS 21*, pages 737–744. 2009.

[25] S. Chib. Marginal likelihood from the gibbs output. *JASA*, 90(432):1313–1321, 1995.

[26] D. Moore, J. Leonard, D. Rus, and S. Teller. Robust distributed network localization with noisy range measurements. In *2nd Int'l Conf. on Emb. Networked Sensor Sys. (SenSys'04)*, pages 50–61, 2004.

